# Low Power Wireless Communication via Reinforcement Learning

**Timothy X Brown**
Electrical and Computer Engineering
University of Colorado
Boulder, CO 80309-0530
timxb@colorado.edu

## Abstract

This paper examines the application of reinforcement learning to a wireless communication problem. The problem requires that channel utility be maximized while simultaneously minimizing battery usage. We present a solution to this multi-criteria problem that is able to significantly reduce power consumption. The solution uses a variable discount factor to capture the effects of battery usage.

## 1 Introduction

Reinforcement learning (RL) has been applied to resource allocation problems in telecommunications, e.g., channel allocation in wireless systems, network routing, and admission control in telecommunication networks [1, 2, 8, 10]. These have demonstrated reinforcement learning can find good policies that significantly increase the application reward within the dynamics of the telecommunication problems. However, a key issue is how to treat the commonly occurring multiple reward and constraint criteria in a consistent way.

This paper will focus on power management for wireless packet communication channels. These channels are unlike wireline channels in that channel quality is poor and varies over time, and often one side of the wireless link is a battery operated device such as a laptop computer. In this environment, power management decides when to transmit and receive so as to simultaneously maximize channel utility and battery life.

A number of power management strategies have been developed for different aspects of battery operated computer systems such as the hard disk and CPU [4, 5]. Managing the channel is different in that some control actions such as shutting off the wireless transmitter make the state of the channel and the other side of the communication unobservable.

In this paper, we consider the problem of finding a power management policy that simultaneously maximizes the radio communication's earned revenue while minimizing battery usage. The problem is recast as a stochastic shortest path problem which in turn is mapped to a discounted infinite horizon with a variable discount factor. Results show significant reductions in power usage.

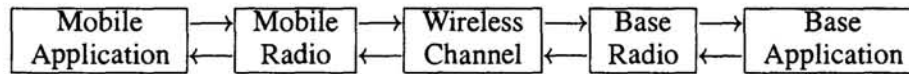

Figure 1: The five components of the radio communication system.

## 2   Problem Description

The problem is comprised of five components as shown in Figure 1: mobile application, mobile radio, wireless channel, base station radio, and base station application. The applications on each end generate packets that are sent via a radio across the channel to the radio and then application on the other side. The application also defines the utility of a given end-to-end performance. The radios implement a simple acknowledgment/retransmit protocol for reliable transmission. The base station is fixed and has a reliable power supply and therefore is not power constrained. The mobile power is limited by a battery and it can choose to turn its radio off for periods of time to reduce power usage. Note that even with the radio off, the mobile system continues to draw power for other uses. The channel adds errors to the packets. The rate of errors depends on many factors such as location of mobile and base station, intervening distance, and levels of interference. The problem requires models for each of these components. To be concrete, the specific models used in this paper are described in the following sections. It should be emphasized that in order to focus on the machine learning issues, simple models have been chosen. More sophisticated models can readily be included.

### 2.1   The Channel

The channel carries fixed-size packets in synchronous time slots. All packet rates are normalized by the channel rate so that the channel carries one packet per unit time in each direction. The forward and reverse channels are orthogonal and do not interfere.

Wireless data channels typically have low error rates. Occasionally, due to interference or signal fading, the channel introduces many errors. This variation is possible even when the mobile and base station are stationary. The channel is modeled by a two state Gilbert-Elliot model [3]. In this model, the channel is in either a "good" or a "bad" state with a packet error probabilities $p_g$ and $p_b$ where $p_g < p_b$. The channel is symmetric with the same loss rate in both directions. The channel stays in each state with a geometrically distributed holding time with mean holding times $h_g$ and $h_b$ time slots.

### 2.2   Mobile and Base Station Application

The traffic generated by the source is a bursty ON/OFF model that alternates between generating no packets and generating packets at rate $r_{ON}$. The holding times are geometrically distributed with mean holding times $h_{ON}$ and $h_{OFF}$. The traffic in each direction is independent and identically distributed.

### 2.3   The Radios

The radios can transmit data from the application and send it on the channel and simultaneously receive data from the other radio and pass it on to its application. The radios implement a simple packet protocol to ensure reliability. Packets from the sources are queued in the radio and sent one by one. Packets consist of a header and data. The header carries acknowledgements (ACK's) with the most recent packet received without error. The header contains a checksum so that errors in the payload can be detected. Errored packets

| Parameter Name | Symbol | Value |
|:---|:---:|:---:|
| Channel Error Rate, Good | $p_g$ | 0.01 |
| Channel Error Rate, Bad | $p_b$ | 0.20 |
| Channel Holding Time, Good | $h_g$ | 100 |
| Channel Holding Time, Bad | $h_b$ | 10 |
| Source On Rate | $r_{\rm ON}$ | 1.0 |
| Source Holding Time, On | $h_{\rm ON}$ | 1 |
| Source Holding Time, Off | $h_{\rm OFF}$ | 10 |
| Power, Radio Off | $P_{\rm OFF}$ | 7 W |
| Power, Radio On | $P_{\rm ON}$ | 8.5 W |
| Power, Radio Transmitting | $P_{\rm TX}$ | 10 W |
| Real Time Max Delay | $d_{max}$ | 3 |
| Web Browsing Time Scale | $d_0$ | 3 |

Table 1: Application parameters.

cause the receiving radio to send a packet with a negative acknowledgment (NACK) to the other radio instructing it to retransmit the packet sequence starting from the errored packet. The NACK is sent immediately even if no data is waiting and the radio must send an empty packet. Only unerrored packets are sent on to the application. The header is assumed to always be received without error[1].

Since the mobile is constrained by power, the mobile is considered the master and the base station the slave. The base station is always on and ready to transmit or receive. The mobile can turn its radio off to conserve power. Every ON-OFF and OFF-ON transition generates a packet with a message in the header indicating the change of state to the base station. These message packets carry no data. The mobile expends power at three levels—$P_{\rm OFF}$, $P_{\rm ON}$, and $P_{tx}$—corresponding to the radio off, receiver on but no packet transmitted, and receiver on packet transmitted.

## 2.4 Reward Criteria

Reward is earned for packets passed in each direction. The amount depends on the application. In this paper we consider three types of applications, an e-mail application, a real-time application, and a web browsing application. In the e-mail application, a unit reward is given for every packet received by the application. In the real time application a unit reward is given for every packet received by the application with delay less than $d_{max}$. The reward is zero otherwise. In the web browsing application, time is important but not critical. The value of a packet with delay $d$ is $(1 - 1/d_0)^d$, where $d_0$ is the desired time scale of the arrivals.

The specific parameters used in this experiment are given in Table 1. These were gathered as typical values from [7, 9]. It should be emphasized that this model is the simplest model that captures the essential characteristics of the problem. More realistic channels, protocols, applications, and rewards can readily be incorporated but for this paper are left out for clarity.

| Component | States |
|---|---|
| Channel | {good,bad} |
| Application | {ON,OFF} |
| Mobile | {ON,OFF} |
| Mobile | {List of waiting and unacknowledged packets and their current delay} |
| Base Station | {List of waiting and unacknowledged packets and their current delay} |

Table 2: Components to System State.

## 3  Markov Decision Processes

At any given time slot, $t$, the system is in a particular configuration, $x$, defined by the state of each of the components in Table 2. The system state is $s = (x, t)$ where we include the time in order to facilitate accounting for the battery. The mobile can choose to toggle its radio between the ON and OFF state and rewards are generated by successfully received packets. The task of the learner is to determine a radio ON/OFF policy that maximizes the total reward for packets received before batteries run out.

The battery life is not a fixed time. First, it depends on usage. Second, for a given drain, the capacity depends on how long the battery was charged, how long it has sat since being charged, the age of the battery, etc. In short, the battery runs out at a random time. The system can be modeled as a stochastic shortest path problem whereby there exists a terminal state, $s_0$, that corresponds to the battery empty in which no more reward is possible and the system remains permanently at no cost.

### 3.1  Multi-criteria Objective

Formally, the goal is to learn a policy for each possible system state so as to *maximize*

$$J^\pi(s) = E\left\{ \left. \sum_{t=0}^{T} c(t) \right| s, \pi \right\},$$

where $E\{\cdot|s,\pi\}$ is the expectation over possible trajectories starting from state $s$ using policy $\pi$, $c(t)$ is the reward for packets received at time $t$, and $T$ is the last time step before the batteries run out.

Typically, $T$ is very large and this inhibits fast learning. So, in order to promote faster learning we convert this problem to a discounted problem that removes the variance caused by the random stopping times. At time $t$, given action $a(t)$, while in state $s(t)$ the terminal state is reached with probability $p_{s(t)}(a(t))$. Setting the value of the terminal state to 0, we can convert our new criterion to maximize:

$$J^\pi(s) = E\left\{ \left. \sum_{t=0}^{\infty} c(t) \prod_{\tau=0}^{t-1} (1 - p_{s(\tau)}(a(\tau))) \right| s, \pi \right\},$$

where the product is the probability of reaching time $t$. In words, future rewards are discounted by $1 - p_s(a)$, and the discounting is larger for actions that drain the batteries faster. Thus a more power efficient strategy will have a discount factor closer to one which correctly extends the effective horizon over which reward is captured.

### 3.2  Q-learning

RL methods solve MDP problems by learning good approximations to the optimal value function, $J^*$, given by the solution to the Bellman optimality equation which takes the

following form:

$$J^*(s) \quad = \quad \max_{a \in A(s)} [E_{s'}\{c(s,a,s') + (1 - p_s(a))J^*(s')\}] \tag{1}$$

where $A(s)$ is the set of actions available in the current state $s$, $c(s,a,s')$ is the effective immediate payoff, and $E_{s'}\{\cdot\}$ is the expectation over possible next states $s'$.

We learn an approximation to $J^*$ using Watkin's Q-learning algorithm. Bellman's equation can be rewritten in Q-factor as

$$J^*(s) \quad = \quad \max_{a \in A(s)} Q^*(s,a) \tag{2}$$

In every time step the following decision is made. The Q-value of turning on in the next state is compared to the Q-value of turning off in the next state. If turning on has higher value the mobile turns on. Else, the mobile turns off.

Whatever our decision, we update our value function as follows: on a transition from state $s$ to $s'$ on action $a$,

$$Q(s,a) \quad = \quad (1-\gamma)Q(s,a) + \gamma \left( c(s,a,s') + (1 - p_s(a)) \max_{b \in A(s')} Q(s',b) \right) \tag{3}$$

where $\gamma$ is the learning rate. In order for Q-learning to perform well, all potentially important state-action pairs $(s,a)$ must be explored. At each state, with probability 0.1 we apply a random action instead of the action recommended by the Q-value. However, we still use (3) to update Q-values using the action $b$ recommended by the Q-values.

### 3.3 Structural Limits to the State Space

For theoretical reasons it is desirable to use a table lookup representation. In practice, since the mobile radio decides using information available to it, this is impossible for the following reasons. The state of the channel is never known directly. The receiver only observes errored packets. It is possible to infer the state, but, only when packets are actually received and channel state changes introduce inference errors.

Traditional packet applications rarely communicate state information to the transport layer. This state information could also be inferred. But, given the quickly changing application dynamics, the application state is often ignored. For the particular parameters in Table 1, (i.e. $r_{ON} = 1.0$) the application is on if and only if it generates a packet so its state is completely specified by the packet arrivals and does not need to be inferred.

The most serious deficiency to a complete state space representation is that when the mobile radio turns OFF, it has no knowledge of state changes in the base station. Even when it is ON, the protocol does not have provisions for transferring directly the state information. Again, this implies that state information must be inferred.

One approach to these structural limits is to use a POMDP approach [6] which we leave to future work. In this paper, we simply learn deterministic policies on features that estimate the state.

### 3.4 Simplifying Assumptions

Beyond the structural problems of the previous section we must treat the usual problem that the state space is huge. For instance, assuming even moderate maximum queue sizes and maximum wait times yields $10^{20}$ states. If one considers e-mail like applications where

| Component | Feature |
|---|---|
| Mobile Radio | is radio ON or OFF |
| Mobile Radio | number of packets waiting at the mobile |
| Mobile Radio | wait time of first packet waiting at the mobile |
| Channel | number of errors received in last 4 time slots |
| Base Radio | number of time slots since mobile was last ON |

Table 3: Decision Features Measured by Mobile Radio

wait times of minutes (1000's of time slot wait times) with many packets waiting possible, the state space exceeds $10^{100}$ states. Thus we seek a representation to reduce the size and complexity of the state space. This reduction is taken in two parts. The first is a feature representation that is possible given the structural limits of the previous section, the second is a function approximation based on these feature vectors.

The feature vectors are listed in Table 3. These are chosen since they are measurable at the mobile radio. For function approximation, we use state aggregation since it provably converges.

## 4 Simulation Results

This section describes simulation-based experiments on the mobile radio control problem. For this initial study, we simplified the problem by setting $p_g = p_b = 0$ (i.e. no channel errors).

State aggregation was used with 4800 aggregate states. The battery termination probability, $p_s(a)$ was simply $P/1000$ where $P$ is the power appropriate for the state and action chosen from Table 1. This was chosen to have an expected battery life much longer than the time scale of the traffic and channel processes.

Three policies were learned, one for each application reward criteria. The resulting policies are tested by simulating for $10^6$ time slots.

In each test run, an upper and lower bound on the energy usage is computed. The upper bound is the case of the mobile radio always on[2]. The lower bound is a policy that ignores the reward criteria but still delivers all the packets. In this policy, the radio is off and packets are accumulated until the latter portion of the test run when they are sent in one large group. Policies are compared using the normalized power savings. This is a measure of how close the policy is to the lower bound with 0% and 100% being the upper and lower bound.

The results are given in Table 4. The table also lists the average reward per packet received by the application. For the e-mail application, which has no constraints on the packets, the average reward is identically one.

## 5 Conclusion

This paper showed that reinforcement learning was able to learn a policy that significantly reduced the power consumption of a mobile radio while maintaining a high application utility. It used a novel variable discount factor that captured the impact of different actions on battery life. This was able to gain 50% to 80% of the possible power savings.

| Application | Normalized Power Savings | Average Reward |
|---|---|---|
| E-mail | 81% | 1 |
| Real Time | 49% | 1.00 |
| Web Browsing | 48% | 0.46 |

Table 4: Simulation Results.

In the application the paper used a simple model of the radio, channel, battery, etc. It also used simple state aggregation and ignored the partially observable aspects of the problem. Future work will address more accurate models, function approximation, and POMDP approaches.

**Acknowledgment**

This work was supported by CAREER Award: NCR-9624791 and NSF Grant NCR-9725778.

## Footnotes

[1]A packet error rate of 20% implies a bit error rate of less than 1%. Error correcting codes in the header can easily reduce this error rate to a low value. The main intent is to simplify the protocol for this paper so that time-outs and other mechanisms do not need to be considered.

[2]There exist policies that exceed this power, e.g. if they toggle ONand OFFoften and generate many notification packets. But, the always on policy is the baseline that we are trying to improve upon.

# References

[1] Boyan, J.A., Littman, M.L., "Packet routing in dynamically changing networks: a reinforcement learning approach," in Cowan, J.D., et al., ed. *Advances in NIPS 6*, Morgan Kauffman, SF, 1994. pp. 671–678.

[2] Brown, T.X, Tong, H., Singh, S., "Optimizing admission control while ensuring quality of service in multimedia networks via reinforcement learning," in *Advances in Neural Information Processing Systems 12*, ed. M. Kearns, et al., MIT Press, 1999, pp. 982–988.

[3] Goldsmith, A.J., Varaiya, P.P., "Capacity, mutual information, and coding for finite state Markov channels," *IEEE T. on Info. Thy.*, v. 42, pp. 868–886, May 1996.

[4] Govil, K., Chan, E., Wasserman, H., "Comparing algorithms for dynamic speed-setting of a low-power cpu," *Proceedings of the First ACM Int. Conf. on Mobile Computing and Networking (MOBICOM)*, 1995.

[5] Helmbold, D., Long, D.D.E., Sherrod, B., "A dynamic disk spin-down technique for mobile computing. *Proceedings of the Second ACM Int. Conf. on Mobile Computing and Networking (MOBICOM)*, 1996.

[6] Jaakola, T., Singh, S., Jordan, M.I., "Reinforcement Learning Algorithm for Partially Observable Markov Decision Problems," in *Advances in Neural Information Processing Systems 7*, ed. G. Tesauro, et al., MIT Press, 1995, pp. 345–352.

[7] Kravits, R., Krishnan, P., "Application-Driven Power Management for Mobile Communication," Wireless Networks, 1999.

[8] Marbach, P., Mihatsch, O., Schulte, M., Tsitsiklis, J.N., "Reinforcement learning for call admission control and routing in integrated service networks," in Jordan, M., et al., ed. *Advances in NIPS 10*, MIT Press, 1998.

[9] Rappaport, T.S., *Wireless Communications: Principles and Practice*, Prentice-Hall Pub., Englewood Cliffs, NJ, 1996.

[10] Singh, S.P., Bertsekas, D.P., "Reinforcement learning for dynamic channel allocation in cellular telephone systems," in *Advances in NIPS 9*, ed. Mozer, M., et al., MIT Press, 1997. pp. 974–980.